# Bayesian Partitioning of Large-Scale Distance Data

**David Adametz**                         **Volker Roth**

Department of Computer Science & Mathematics
University of Basel
Basel, Switzerland
{david.adametz,volker.roth}@unibas.ch

## Abstract

A Bayesian approach to partitioning distance matrices is presented. It is inspired by the *Translation-invariant Wishart-Dirichlet* process (TIWD) in [1] and shares a number of advantageous properties like the fully probabilistic nature of the inference model, automatic selection of the number of clusters and applicability in semi-supervised settings. In addition, our method (which we call *fastTIWD*) overcomes the main shortcoming of the original TIWD, namely its high computational costs. The fastTIWD reduces the workload in each iteration of a Gibbs sampler from $O(n^3)$ in the TIWD to $O(n^2)$. Our experiments show that the cost reduction does not compromise the quality of the inferred partitions. With this new method it is now possible to 'mine' large relational datasets with a probabilistic model, thereby automatically detecting new and potentially interesting clusters.

## 1   Introduction

In cluster analysis we are concerned with identifying subsets of $n$ objects that share some similarity and therefore potentially belong to the same sub-population. Many practical applications leave us without direct access to vectorial representations and instead only supply pairwise distance measures collected in a matrix $D$. This poses a serious challenge, because great parts of geometric information are hereby lost that could otherwise help to discover hidden structures. One approach to deal with this is to encode geometric invariances in the probabilistic model, as proposed in [1]. The most important properties that distinguish this *Translation-invariant Wishart-Dirichlet Process* (TIWD) from other approaches working on pairwise data are its fully probabilistic model, automatic selection of the number of clusters, and its applicability in semi-supervised settings in which not all classes are known in advance. Its main drawback, however, is the high computational cost of order $O(n^3)$ per sweep of a Gibbs sampler, limiting its applicability to relatively small data sets.

In this work we present an alternative method which shares all the positive properties of the TIWD while reducing the computational workload to $O(n^2)$ per Gibbs sweep. In analogy to [1] we call this new approach *fastTIWD*. The main idea is to solve the problem of missing geometric information by a normalisation procedure, which chooses one particular geometric embedding of the distance data and allows us to use a simple probabilistic model for inferring the unknown underlying partition. The construction we use is guaranteed to give the optimal such geometric embedding if the true partition was known. Of course, this is only a hypothetical precondition, but we show that even rough prior estimates of the true partition significantly outperform 'naive' embedding strategies. Using a simple hierarchical clustering model to produce such prior estimates leads to clusterings being at least of the same quality as those obtained by the original TIWD. The algorithmic contribution here is an efficient algorithm for performing this normalisation procedure in $O(n^2)$ time, which makes the whole pipeline from distance matrix to inferred partition an $O(n^2)$ process (assuming a constant number of Gibbs sweeps). Detailed complexity analysis shows not only a worst-case complexity reduction from $O(n^3)$ to $O(n^2)$, but also a drastic speed improvement. We demonstrate

this performance gain for a dataset containing $\approx 350$ clusters, which now can be analysed in 6 hours instead of $\approx 50$ days with the original TIWD.

It should be noted that both the TIWD and our fastTIWD model expect (squared) Euclidean distances on input. While this might be seen as a severe limitation, we argue that (i) a 'zoo' of Mercer kernels has been published in the last decade, e.g. kernels on graphs, sequences, probability distributions etc. All these kernels allow the construction of squared Euclidean distances; (ii) efficient preprocessing methods like randomised versions of kernel PCA have been proposed, which can be used to transform an initial matrix into one of squared Euclidean type; (iii) one might even use an arbitrary distance matrix hoping that the resulting model mismatch can be tolerated.

In the next section we introduce a probabilistic model for partitioning inner product matrices, which is generalised in section 3 to distance matrices using a preprocessing step that breaks the geometric symmetry inherent in distance representations. Experiments in section 4 demonstrate the high quality of clusterings found by our method and its superior computational efficiency over the TIWD.

## 2  A Wishart Model for Partitioning Inner Product Matrices

Suppose there is a matrix $X \in \mathbb{R}^{n \times d}$ representing $n$ objects in $\mathbb{R}^d$ that belong to one of $k$ sub-populations. For identifying the underlying cluster structure, we formulate a generative model by assuming the columns $\boldsymbol{x}_i \in \mathbb{R}^n$, $i = 1 \ldots d$ are i.i.d. according to a normal distribution with zero mean and covariance $\Sigma_{n \times n}$, i.e. $\boldsymbol{x}_i \sim \mathcal{N}(\mathbf{0}_n, \Sigma)$, or in matrix notation: $X \sim \mathcal{N}(0_{n \times d}, \Sigma \otimes I)$. Then, $S = \frac{1}{d} X X^t \in \mathbb{R}^{n \times n}$ is central Wishart distributed, $S \sim \mathcal{W}_d(\Sigma)$. For convenience we define the *generalised* central Wishart distribution which also allows rank-deficient $S$ and/or $\Sigma$ as

$$p(S|\Psi, d) \propto \det(S)^{\frac{1}{2}(d-n-1)} \det(\Psi)^{\frac{d}{2}} \exp\left[-\frac{d}{2} \operatorname{tr}(\Psi S)\right], \tag{1}$$

where $\det(\bullet)$ is the product of non-zero eigenvalues and $\Psi$ denotes the (generalised) inverse of $\Sigma$. The likelihood as a function in $\Psi$ is

$$\mathcal{L}(\Psi) = \det(\Psi)^{\frac{d}{2}} \exp\left[-\frac{d}{2} \operatorname{tr}(\Psi S)\right]. \tag{2}$$

Consider now the case where we observe $S$ without direct access to $X$. Then, an orthogonal transformation $X \leftarrow OX$ cannot be retrieved anymore, but it is reasonable to assume such rotations are irrelevant for finding the partition. Following the Bayesian inference principle, we complement the likelihood with a prior over $\Psi$. Since by assumption there is an underlying joint normal distribution, a zero entry in $\Psi$ encodes conditional independence between two objects, which means that block diagonal $\Psi$ matrices define a suitable partitioning model in which the joint normal is decomposed into independent cluster-wise normals. Note that the inverse of a block diagonal matrix is also block diagonal, so we can formulate the prior in terms of $\Sigma$, which is easier to parametrise. For this purpose we adapt the method in [2] using a Multinomial-Dirichlet process model [3, 4, 5] to define a flexible prior distribution over block matrices without specifying the exact number of blocks. We only briefly sketch this construction and refer the reader to [1, 2] for further details. Let $\mathbb{B}_n$ be the set of partitions of the index set $[n]$. A partition $B \in \mathbb{B}_n$ can be represented in matrix form as $B(i,j) = 1$ if $y(i) = y(j)$ and $B(i,j) = 0$ otherwise, with $y$ being a function that maps $[n]$ to some label set $\mathbb{L}$. Alternatively, $B$ may be represented as a set of disjoint non-empty subsets called 'blocks' $b$. A *partition process* is a series of distributions $P_n$ on the set $\mathbb{B}_n$ in which $P_n$ is the marginal of $P_{n+1}$. Using a multinomial model for the labels and a Dirichlet prior with rate parameter $\xi$ on the mixing proportions, we may integrate out the latter and derive a Dirichlet-Multinomial prior over labels. Finally, after using a 'label forgetting' transformation, the prior over $B$ is:

$$p(B|\xi, k) = \frac{k!}{(k - k_B)!} \frac{\Gamma(\xi) \prod_{b \in B} \Gamma(n_b + \xi/k)}{[\Gamma(\xi/k)]^{k_B} \Gamma(n + \xi)}. \tag{3}$$

In this setting, $k$ is the number of blocks in the population ($k$ can be infinite, which leads to the Ewens Process [6], a.k.a. Chinese Restaurant Process), $n_b$ is the number of objects in block $b$ and $k_B \leq k$ is the total number of blocks in $B$. The prior is *exchangeable* meaning rows and columns can be (jointly) permuted arbitrarily and therefore partition matrices can always be brought to block diagonal form. To specify the variances of the normal distributions, the models in [1, 2] use two global parameters, $\alpha, \beta$, for the within- and between-class scatter. This model can be easily extended to include block-wise scatter parameters, but for the sake of simplicity we will stay with the simple parametrisation here. The final block diagonal covariance matrix used in (2) has the form

$$\Sigma = \Psi^{-1} = \alpha(I_n + \theta B), \quad \text{with} \quad \theta := \beta/\alpha. \tag{4}$$

**Inference by way of Gibbs sampling.** Multiplying the Wishart likelihood (2), the prior over partitions (3) and suitable priors over $\alpha, \theta$ gives the joint posterior. Inference for $B$, $\alpha$ and $\theta$ can then be carried out via a Gibbs sampler. Each Gibbs sweep can be efficiently implemented since both trace and determinant in (2) can be computed analytically, see [1]:

$$\mathrm{tr}(\Psi S) = \sum_{b \in B} \frac{1}{\alpha} \left[ \mathrm{tr}(S_{bb}) - \frac{\theta}{1 + n_b \theta} \bar{S}_{bb} \right] = \frac{1}{\alpha} \left[ \mathrm{tr}(S) - \sum_{b \in B} \frac{\theta}{1 + n_b \theta} \bar{S}_{bb} \right], \qquad (5)$$

where $S_{bb}$ denotes the block submatrix corresponding to the $b$th diagonal block in $B$, and $\bar{S}_{bb} = \mathbb{1}_b^t S_{bb} \mathbb{1}_b$. $\mathbb{1}_b$ is the *indicator function* mapping block $b$ to a $\{0,1\}^n$ vector, whose elements are 1 if a sample is contained in $b$, or 0 otherwise. For the determinant one derives

$$\det(\Psi) = \alpha^{-n} \prod_{b \in B} (1 + \theta n_b)^{-1}. \qquad (6)$$

The conditional likelihood for $\alpha$ is Inv-Gamma$(r, s)$ with shape parameter $r = n \cdot d/2 - 1$ and scale $s = \frac{d}{2} \left[ \mathrm{tr}(S) - \sum_{b \in B} \frac{\theta}{1 + n_b \theta} \bar{S}_{bb} \right]$. Using the prior $\alpha \sim$ Inv-Gamma$(r_0 \cdot d/2, s_0 \cdot d/2)$, the posterior is of the same functional form, and we can integrate out $\alpha$ analytically:

$$P_n(B|\bullet) \propto P_n(B|\xi, k) \det(\Psi)_{(\alpha=1)}^{d/2} \left[ \frac{d}{2} \big( \mathrm{tr}(\Psi S)_{(\alpha=1)} + s_0 \big) \right]^{-(n+r_0)d/2}, \qquad (7)$$

where $\det(\Psi)_{(\alpha=1)} = \prod_{b \in B} (1 + \theta n_b)^{-1}$ and $\mathrm{tr}(\Psi S)_{(\alpha=1)} = \mathrm{tr}(S) - \sum_{b \in B} \frac{\theta}{1 + n_b \theta} \bar{S}_{bb}$. Note that the (usually unknown) degree of freedom $d$ has the formal role of an annealing parameter, and it can indeed be used to 'cool' the Markov chain by increasing $d$, if desired, until a partition is 'frozen'.

**Complexity analysis.** In one sweep of the Gibbs sampler, we have to iteratively compute the membership probability of one object indexed by $i$ to the $k_B$ currently existing blocks in partition $B$ (plus one new block), given the assignments for the $n-1$ remaining ones denoted by the superscript $^{(-i)}$ [7, 8]. In every step of this inner loop over $k_B$ existing blocks we have to evaluate the Wishart likelihood, i.e. trace (5) and determinant (6). Given trace $\mathrm{tr}^{(-i)}$, we update $\bar{S}_{bb}$ for $k_B$ blocks $b \in B$ which in total needs $O(n)$ operations. Given $\det^{(-i)}$, the computation of all $k_B$ updated determinants induces costs of $O(k_B)$. In total, there are $n$ objects, so a full sweep requires $O(n^2 + nk_B)$ operations, which is equal to $O(n^2)$ since the maximum number of blocks is $n$, i.e. $k_B \leq n$. Following [1], we update $\theta$ on a discretised grid of values which adds $O(k_B)$ to the workload, thus not changing the overall complexity of $O(n^2)$. Compared to the original TIWD, the worst case complexity in the Dirichlet process model with an infinite number of blocks in the population, $k = \infty$, is reduced from $O(n^3)$ to $O(n^2)$.

## 3 The fastTIWD Model for Partitioning Distance Matrices

Consider now the case where $S$ is not accessible, but only squared pairwise distances $D \in \mathbb{R}^{n \times n}$:

$$D(i, j) = S(i, i) + S(j, j) - 2 S(i, j). \qquad (8)$$

Observing one specific $D$ does not imply a unique corresponding $S$, since there is a surjective mapping from a set of $S$-matrices to $D$, $\mathbb{S}(D) \mapsto D$. Hereby, not only do we lose information about orthogonal transformations of $X$, but also information about the origin of the coordinate system. If $S_*$ is one (any) matrix that fulfills (8) for a specific $D$, the set $\mathbb{S}(D)$ is formally defined as $\mathbb{S} = \{ S | S = S_* + \mathbf{1} v^t + v \mathbf{1}^t, S \succeq 0, v \in \mathbb{R}^n \}$ [9]. The Wishart distribution, however, is not invariant against the choice of $S \in \mathbb{S}$. In fact, if $S_* \sim \mathcal{W}(\Sigma)$, the distribution of a general $S \in \mathbb{S}$ is *non-central* Wishart, which can be easily seen as follows: $\mathbb{S}$ is exactly the set of inner product matrices that can be constructed by varying $c \in \mathbb{R}^d$ in a modified matrix normal model $X \sim \mathcal{N}(M, \Sigma \otimes I_d)$ with mean matrix $M = \mathbf{1}_n c^t$. Note that now the $d$ columns in $X$ are still independent, but no longer identically distributed. Note further that 'shifts' $c_i$ do not affect pairwise distances between rows in $X$. The modified matrix normal distribution implies that $S = \frac{1}{d} X X^t$ is *non-central* Wishart, $S \sim \mathcal{W}(\Sigma, \Theta)$, with non-centrality matrix $\Theta := \Sigma^{-1} M M^t$. The practical use, however, is limited by its complicated form and the fundamental problem of estimating $\Theta$ based on only one single observation $S$. It is thus desirable to work with a simpler probabilistic model. In principle, there are two possibilities: either the likelihood is reformulated as being constant over all $S \in \mathbb{S}$ (the approach taken in [1], called the *translation-invariant Wishart distribution*), or one tries to find a 'good' candidate matrix $S'_*$ that is 'close' to the underlying $S_*$ and uses the much

simpler central Wishart model. Both approaches have their pros and cons: encoding the translation invariance directly in the likelihood is methodologically elegant and seems to work well in a couple of experiments (cf. [1]), but it induces high computational cost. The alternative route of searching for a good candidate $S'_*$ close to $S_*$ is complicated, because $S_*$ is unknown and it is not immediately clear what 'close' means. The positive aspect of this approach is the heavily reduced computational cost due to the formal simplicity of the central Wishart model. It is important to discuss the 'naive' way of finding a good candidate $S'_*$ by subtracting the empirical column means in $X$, thus removing the shifts $c_i$. This normalisation procedure can be implemented solely based on $S$, leading to the well-known centering procedure in kernel PCA, [10]:

$$S_c = Q_I S\, Q_I^t, \quad \text{with projection} \quad Q_I = I - (1/n)\mathbf{1}\mathbf{1}^t. \tag{9}$$

Contrary to the PCA setting, however, this column normalisation induced by $Q_I$ does not work well here, because the elements of a column vector in $X$ are *not* independent. Rather, they are coupled via the $\Sigma$ component in the covariance tensor $\Sigma \otimes I_d$. Hereby, we not only remove the shifts $c_i$, but also alter the distribution: the non-centrality matrix does not vanish in general and as a result, $S_c$ is no longer central Wishart distributed.

In the following we present a solution to the problem of finding a candidate matrix $S'_*$ that recasts inference based on the translation-invariant Wishart distribution as a method to reconstruct the optimal $S_*$. Our proposal is guided by a particular analogy between trees and partition matrices and aims at exploiting a tree-structure to guarantee low computational costs. The construction has the same functional form as (9), but uses a different projection matrix $Q$.

**The translation-invariant Wishart distribution.** Let $S_*$ induce pairwise distances $D$. Assuming that $S_* \sim \mathcal{W}_d(\Sigma)$, the distribution of an arbitrary member $S \in \mathbb{S}(D)$ can be derived analytically as a generalised central Wishart distribution with a rank-deficient covariance, see [2]. Its likelihood in the rank-deficient inverse covariance matrix $\widetilde{\Psi}$ is

$$\mathcal{L}(\widetilde{\Psi}) \propto \det(\widetilde{\Psi})^{\frac{d}{2}} \exp\big[ -\tfrac{d}{2}\mathrm{tr}(\widetilde{\Psi}S_*)\big] = \det(\widetilde{\Psi})^{\frac{d}{2}} \exp\big[\tfrac{d}{4}\mathrm{tr}(\widetilde{\Psi}D)\big], \tag{10}$$

with $\widetilde{\Psi} = \Psi - (\mathbf{1}^t\Psi\mathbf{1})^{-1}\Psi\mathbf{1}\mathbf{1}^t\Psi$. Note that although $S_*$ appears in the first term in (10), the density is constant on all $S \in \mathbb{S}(D)$, meaning it can be replaced by any other member of $\mathbb{S}(D)$. Note further that $\mathbb{S}$ also contains rank-deficient matrices (like, e.g. the column normalised $S_c$). By multiplying (10) with the product of nonzero eigenvalues of such a matrix raised to the power of $(d - n - 1)/2$, a valid generalised central Wishart distribution is obtained (see (1)), which is normalised on the manifold of positive semi-definite matrices of rank $r = n - 1$ with $r$ distinct positive eigenvalues [11, 12, 13]. Unfortunately, (10) has a simple form only in $\widetilde{\Psi}$, but not in the original $\Psi$, which finally leads to the $O(n^3)$ complexity of the TIWD model.

**Selecting an optimal candidate $S_*$.** Introducing the projection matrix

$$Q = I - \tfrac{1}{\mathbf{1}^t\Psi\mathbf{1}}\mathbf{1}\mathbf{1}^t\Psi, \tag{11}$$

one can rewrite $\widetilde{\Psi}$ in (10) as $\Psi Q$ or, equivalently, as $Q^t\Psi Q$, see [2] for details. Assume now $S \sim \mathcal{W}_d(\Sigma)$ induces distances $D$ and consider the transformed $S_* = QSQ^t$. Note that this transformation does not change the distances, i.e. $S \in \mathbb{S}(D) \Leftrightarrow S_* \in \mathbb{S}(D)$, and that $QSQ^t$ has rank $r = n - 1$ (because $Q$ is a projection with kernel $\mathbf{1}$). Plugging our specific $S_* = QSQ^t$ into (10), extending the likelihood to a generalised central Wishart (1) with rank-deficient inverse covariance $\widetilde{\Psi}$, exploiting the identity $QQ = Q$ and using the the cyclic property of the trace, we arrive at

$$p(QSQ^t|\widetilde{\Psi}, d) \propto \det(QSQ^t)^{\frac{1}{2}(d-n-1)} \det(\widetilde{\Psi})^{\frac{d}{2}} \exp\big[ -\tfrac{d}{2}\,\mathrm{tr}(\Psi QSQ^t)\big]. \tag{12}$$

By treating $Q$ as a fixed matrix, this expression can also be seen as a central Wishart in the transformed matrix $S_* = QSQ^t$, parametrised by the full-rank matrix $\Psi$ if $\det(\widetilde{\Psi})$ is substituted by the appropriate normalisation term $\det(\Psi)$. From this viewpoint, inference using the translation-invariant Wishart distribution can be interpreted as finding a (rank-deficient) representative $S_* = QSQ^t \in \mathbb{S}(D)$ which follows a generalised central Wishart distribution with full-rank inverse covariance matrix $\Psi$. For inferring $\Psi$, the rank deficiency of $S_*$ is not relevant, since only the likelihood is needed. Thus $S_*$ can be seen as an optimal candidate inner-product matrix in the set $\mathbb{S}(D)$ for a *central* Wishart model parametrised by $\Psi$.

**Approximating $S_*$ with trees.** The above selection of $S_* \in \mathbb{S}(D)$ cannot be directly used in a constructive way, since $Q$ in (11) depends on unknown $\Psi$. If, on the other hand, we had some initial estimate of $\Psi$, we could find a reasonable transformation $Q'$ and hereby a reasonable candidate $S'_*$. Note that even if the estimate of $\Psi$ is far away from the true inverse covariance, the pairwise distances are at least guaranteed not to change under $Q'S(Q')^t$.

One particular estimate would be to assume that every object forms a singleton cluster, which means that our estimate of $\Psi$ is an identity matrix. After substitution into (11) it is easily seen that this assumption results in the column-normalisation projection $Q_I$ defined in (9). However, if we assume that there is some non-trivial cluster structure in the data, this would be a very poor approximation. The main difficulty in finding a better estimate is to not specify the number of blocks. Our construction is guided by an analogy between binary trees and weighted sums of cut matrices, which are binary complements of partition matrices with two blocks. We use a binary tree with $n$ leaves representing $n$ objects. It encodes a path distance matrix $D_{\text{tree}}$ between those $n$ objects, and for an optimal tree $D_{\text{tree}} = D$. Such an optimal tree exists only if $D$ is additive, and the task of finding an approximation is a well-studied problem. We will not discuss the various tree reconstruction algorithms, but only mention that there exist algorithms for reconstructing the closest ultrametric tree (in the $\ell_\infty$ norm) in $O(n^2)$ time, [14].

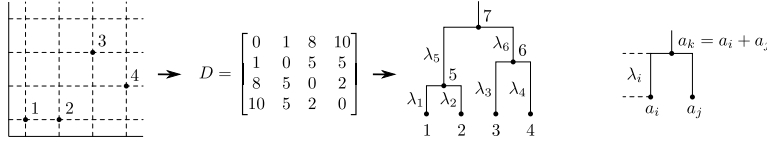

Figure 1: From left to right: Unknown samples $X$, pairwise distances collected in $D$, closest tree structure and an exemplary building block.

A tree metric induced by $D_{\text{tree}}$ is composed of elementary cut (pseudo-)metrics. Any such metric lies in the metric space $L_1$ and is also a member of $(L_2)^2$, which is the metric part of the space of squared Euclidean distance matrices $D$. Thus, there exists a positive (semi-)definite $S_{\text{tree}}$ such that $(D_{\text{tree}})_{ij} = (S_{\text{tree}})_{ii} + (S_{\text{tree}})_{jj} - 2(S_{\text{tree}})_{ij}$. In fact, any matrix $S_{\text{tree}}$ has a canonical decomposition into a weighted sum of 2-block partition matrices, which is constructed by cutting all edges ($2n - 2$ for a rooted tree) and observing the resulting classification of leaf nodes. Suppose, we keep track of such an assignment with indicator $\mathbb{1}_j$ induced by a single cut $j$, then the inner product matrix is

$$S_{\text{tree}} = \sum_{j=1}^{2n-2} \lambda_j (\mathbb{1}_j \mathbb{1}_j^t + \bar{\mathbb{1}}_j \bar{\mathbb{1}}_j^t), \tag{13}$$

where $\lambda_j$ is the weight of edge $j$ to be cut and $\bar{\mathbb{1}}_j \mapsto \{0,1\}^n$ is the complementary assignment, i.e. $\mathbb{1}_j$ flipped. Each term $(\mathbb{1}_j \mathbb{1}_j^t + \bar{\mathbb{1}}_j \bar{\mathbb{1}}_j^t)$ is a 2-block partition matrix. We demonstrate the construction of $S_{\text{tree}}$ in Fig. 2 for a small dataset of $n = 25$ objects sampled from $S \sim \mathcal{W}_d(\Sigma)$ with $d = 25$ and $\Sigma = \alpha(I_n + \theta B)$ as defined in (4) with $\alpha = 2$ and $\theta = 1$. $B$ contains 3 blocks and is depicted in the first panel. The remaining panels show the single-linkage clustering tree, all $2n - 2 = 48$ weighted 2-block partition matrices, and the final $S_{\text{tree}}$ (= sum of all individual 2-block matrices, rescaled to full gray-value range). Note that single-linkage fails to identify the clusters in the three branches closest to root, but still the structure of $B$ is clearly visible in $S_{\text{tree}}$.

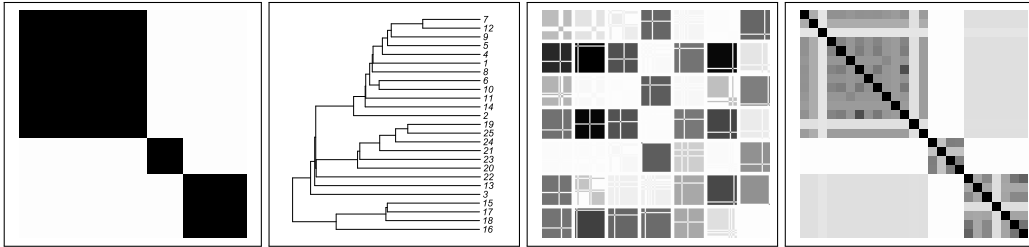

Figure 2: Inner product matrix of a tree. Left to right: Partition matrix $B$ for $n = 25$ objects in 3 clusters, single-linkage tree, all weighted 2-block partition matrices, final $S_{\text{tree}}$.

The idea is now to have $S_{\text{tree}}$ as an estimate of $\Sigma$, and use its inverse $\Psi_{\text{tree}}$ to construct $Q_{\text{tree}}$ in (11), which, however, naively would involve an $O(n^3)$ Cholesky decomposition of $S_{\text{tree}}$.

**Theorem 1.** *The $n \times n$ matrix $S_* = Q_{tree} S Q_{tree}^t$ can be computed in $O(n^2)$ time.*

For the proof we need the following lemma:

**Lemma 1.** *The product of $S_{tree} \in \mathbb{R}^{n \times n}$ and a vector $\mathbf{y} \in \mathbb{R}^n$ can be computed in $O(n)$ time.*

*Proof.* (of lemma 1) Restating (13) and defining $m := 2n - 2$, we have

$$
\begin{aligned}
S_{\text{tree}}\mathbf{y} &= \sum_{j=1}^{m} \lambda_j \big(\mathbb{1}_j \mathbb{1}_j^t + \bar{\mathbb{1}}_j \bar{\mathbb{1}}_j^t\big)\mathbf{y} = \sum_{j=1}^{m} \lambda_j \big(\mathbb{1}_j \sum_{l=1}^{n} 1_{jl}\, y_l + \bar{\mathbb{1}}_j \sum_{l=1}^{n} \bar{1}_{jl}\, y_l\big) \\
&= \sum_{l=1}^{n} y_l \sum_{j=1}^{m} \lambda_j \bar{\mathbb{1}}_j + \sum_{j=1}^{m} \lambda_j \mathbb{1}_j \sum_{l=1}^{n} 1_{jl} y_l - \sum_{j=1}^{m} \lambda_j \bar{\mathbb{1}}_j \sum_{l=1}^{n} 1_{jl} y_l.
\end{aligned}
\tag{14}
$$

In the next step, let us focus specifically on the $i$th element of the resulting vector. Furthermore, assume $R_i$ is the set of all nodes on the branch starting from node $i$ and leading to the tree's root:

$$
\begin{aligned}
(S_{\text{tree}}\mathbf{y})_i &= \sum_{l=1}^{n} y_l \sum_{j \notin R_i} \lambda_j + \sum_{j \in R_i} \big(\lambda_j \sum_{l \in R_j} y_l\big) - \sum_{j \notin R_i} \big(\lambda_j \sum_{l \in R_j} y_l\big) \\
&= \sum_{l=1}^{n} y_l \big(\sum_{j=1}^{m} \lambda_j - \sum_{j \in R_i} \lambda_j\big) + 2 \sum_{j \in R_i} \lambda_j y_j - \sum_{j=1}^{m} \lambda_j y_j.
\end{aligned}
\tag{15}
$$

Note that $\sum_{l=1}^{n} y_l$, $\sum_{j=1}^{m} \lambda_j$ and $\sum_{j=1}^{m} \lambda_j y_j$ are constants and computed in $O(n)$ time. For each element $i$, we are now left to find $R_i$ in order to determine the remaining two terms. This can be done directly on the tree structure in two separate traversals:

1. Bottom up: Starting from the leaf nodes, store the sum of both childrens' $y$ values in their parent node $j$ (see Fig. 1, rightmost), then ascend. Do the same for $\lambda_j$ and compute $\lambda_j y_j$.
2. Top down: Starting from the root node, recursively descend into the child nodes $j$ and sum up $\lambda_j$ and $\lambda_j y_j$ until reaching the leafs. This implicitly determines $R_i$.

It is important to stress that the above two tree traversals fully describe the complete algorithm. $\square$

*Proof.* (of theorem 1) First, note that only the matrix-vector product $\mathbf{a} := \Psi_{\text{tree}}\mathbf{1}$ is needed in

$$
\begin{aligned}
Q_{\text{tree}} S Q_{\text{tree}}^t &= \big(I - \tfrac{1}{\mathbf{1}^t \Psi_{\text{tree}} \mathbf{1}} \mathbf{1}\mathbf{1}^t \Psi_{\text{tree}}\big) S \big(I - \Psi_{\text{tree}} \tfrac{1}{\mathbf{1}^t \Psi_{\text{tree}} \mathbf{1}} \mathbf{1}\mathbf{1}^t\big) \\
&= S - (1/\mathbf{1}^t \mathbf{a})\, \mathbf{1}\mathbf{a}^t S - (1/\mathbf{1}^t \mathbf{a})\, S\, \mathbf{a}\mathbf{1}^t + (1/\mathbf{1}^t \mathbf{a})^2\, \mathbf{1}\mathbf{a}^t S\, \mathbf{a}\mathbf{1}^t.
\end{aligned}
\tag{16}
$$

One way of computing $\mathbf{a} = \Psi_{\text{tree}}\mathbf{1}$ is to employ conjugate gradients (CG) and iteratively minimise $||S_{\text{tree}}\mathbf{a} - \mathbf{1}||^2$. Theoretically, CG is guaranteed to find the true $\mathbf{a}$ in $O(n)$ iterations, each evaluating one matrix-vector product $(S_{\text{tree}}\mathbf{y})$, $\mathbf{y} \in \mathbb{R}^n$. Due to lemma 1, $\mathbf{a}$ can be computed in $O(n^2)$ time and is used in (16) to compute $S_* = Q_{\text{tree}} S Q_{\text{tree}}^t$ (only matrix-vector products, so $O(n^2)$ complexity is maintained). $\square$

## 4 Experiments

**Synthetic examples: normal clusters.** In a first experiment we investigate the performance of our method on artificial datasets generated in accordance with underlying model assumptions. A partition matrix $B$ of size $n = 200$ containing $k = 3$ blocks is sampled from which we construct $\Sigma_B = \alpha(I + \theta B)$. Then, $X$ is drawn from $\mathcal{N}(M = 40 \cdot \mathbf{1}_n \mathbf{1}_d^t, \Sigma = \Sigma_B \otimes I_d)$ with $d = 300$ to generate $S = \frac{1}{d} XX^t$ and $D$. The covariance parameters are set to $\alpha = 2$ and $\theta = 15/d$, which defines a rather difficult clustering problem with a hardly visible structure in $D$ as can be seen in the left part of Fig. 3. We compared the method to three different hierarchical clustering strategies (single-linkage, complete-linkage, Ward's method), to the standard central Wishart model using two different normalisations of $S$ ('WD_C': column normalisation using $S_c = Q_I S Q_I^t$ and 'WD_R': additional row normalisation after embedding $S_c$ using kernel PCA) and to the original TIWD model. The experiment was repeated 200 times and the quality of the inferred clusters was measured by the adjusted Rand index w.r.t. the true labels. For the hierarchical methods we report two different performance values: splitting the tree such that the 'true' number $k = 3$ of clusters is obtained and computing the best value among all possible splits into $[2, n]$ clusters ('*.best' in the boxplot). The reader should notice that both values are in favour of the hierarchical algorithms, since neither the true $k$ nor the true labels are used for inferring the clusters in the Wishart-type methods. From the right part of Fig. 3 we conclude that (i) both 'naive' normalisation strategies WD_C and WD_R are clearly outperformed by TIWD and fastTIWD ('fTIWD' in the boxplot). Significance of pairwise performance differences is measured with a nonparametric Kruskal-Wallis test with a

Bonferroni-corrected post-test of Dunn's type, see the rightmost panel; (ii) the hierarchical methods have severe problems with high dimensionality and low class separation, and optimising the tree cutting does not help much. Even Ward's method (being perfectly suited for spherical clusters) has problems; (iii) there is no significant difference between TIWD and fastTIWD.

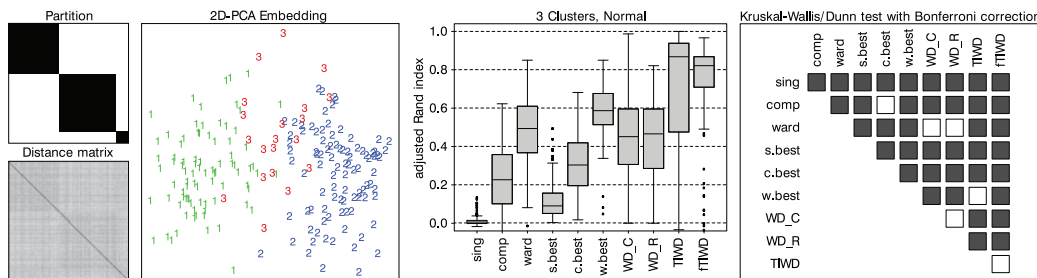

Figure 3: Normal distributed toy data. Left half: Partition matrix (top), distance matrix (bottom) and 2D-PCA embedding of a dataset drawn from the generative model. Right half: Agreement with 'true' labels measured by the adjusted Rand index (left) and outcome of a Kruskal-Wallis/Dunn test (right). Black squares mean two methods are different at a 'family' $p$-value $\leq 0.05$.

**Synthetic examples: log-normal clusters.** In a second toy example we explicitly violate underlying model assumptions. For this purpose we sample again 3 clusters in $d = 300$ dimensions, but now use a log-normal distribution that tends to produce a high number of 'atypical' samples. Note that such a distribution should not induce severe problems for hierarchical methods when optimising the Rand index over all possible tree cuttings, since the 'atypical' samples are likely to form singleton clusters while the main structure is still visible in other branches of the tree. This should be particularly true for Ward's method, since we still have spherically shaped clusters. As for the fastTIWD model, we want to test if the prior over partitions is flexible enough to introduce additional singleton clusters: In the experiment, it performed at least as well as Ward's method, and clearly outperformed single- and complete-linkage. We also compared it to the affinity-propagation method (AP), which, however, has severe problems on this dataset, even when optimising the *input preference* parameter that affects the number of clusters in the partition.

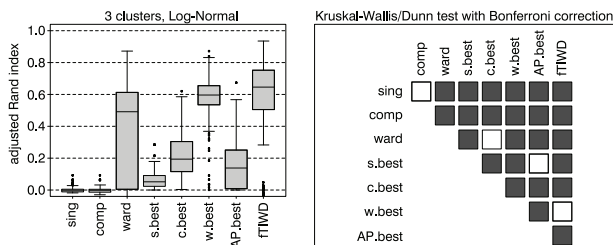

Figure 4: Log-normal distributed toy data. Left: Agreement with 'true' labels measured by the adjusted Rand index. Right: Outcome of a Kruskal-Wallis/Dunn test, analogous to Fig. 3.

**Semi-supervised clustering of protein sequences.** As large-scale application we present a semi-supervised clustering example which is an upscaled version of an experiment with protein sequences presented in [1]. While traditional semi-supervised classifiers assume at least one labelled object per class, our model is flexible enough to allow additional new clusters that have no counterpart in the subset of labelled objects. We apply this idea on two different databases, one being high quality due to manual annotation with a stringent review process (SwissProt) while the other contains automatically annotated proteins and is not reviewed (TrEMBL). The annotations in SwissProt are used as supervision information resulting in a set of class labels, whereas the proteins in TrEMBL are treated as unlabelled objects, potentially forming new clusters. In contrast to a relatively small set of *globin* sequences in [1], we extract a total number of 12,290 (manually or automatically) annotated proteins to have some role in oxygen transport or binding. This set contains a richer class including, for instance, *hemocyanins*, *hemerythrins*, *chlorocruorins* and *erythrocruorins*.

The proteins are represented as a matrix of pairwise alignment scores. A subset of 1731 annotated sequences is from SwissProt, resulting in 356 protein classes. Among the 10,559 TrEMBL sequences

we could identify 23 new clusters which are dissimilar to any SwissProt proteins, see Fig. 5. Most of the newly identified clusters contain sequences sharing some rare and specific properties. In accordance with the results in [1], we find a large new cluster containing *flavohemoglobins* from specific species of funghi and bacteria that share a certain domain architecture composed of a globin domain fused with ferredoxin reductase-like FAD- and NAD-binding modules. An additional example is a cluster of proteins with *chemotaxis methyl-accepting receptor* domain from a very special class of *magnetic* bacteria to orient themselves according to earth's magnetic field. The domain architecture of these proteins involving 6 domains is unique among all sequences in our dataset. Another cluster contains *iron-sulfur cluster repair di-iron proteins* that build on a polymetallic system, the *di-iron center*, constituted by two iron ions bridged by two sulfide ions. Such di-iron centers occur only in this new cluster.

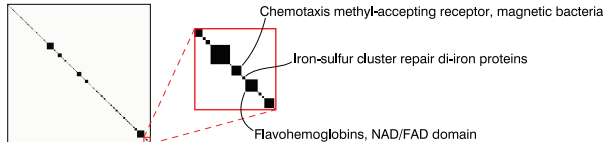

Figure 5: Partition of all 12,290 proteins into 379 clusters: 356 predefined by sequences from SwissProt and 23 new formed by sequences from TrEMBL (red box).

In order to gain the above results, 5000 Gibbs sweeps were conducted in a total runtime of $\approx 6$ hours. Although section 2 highlighted the worst-case complexity of the original TIWD, it is also important to experimentally compare both models in a real world scenario: we ran 100 sweeps with each fastTIWD and TIWD and hereby observed an average improvement of factor 192, which would lead to an estimated runtime of 1152 hours ($\approx 50$ days) for the latter model. On a side note, automatic cluster identification is a nice example for benefits of large-scale data mining: clearly, one could theoretically also identify special sequences by digging into various protein domain databases, but without precise prior knowledge, this would hardly be feasible for $\approx 12,000$ proteins.

## 5 Conclusion

We have presented a new model for partitioning pairwise distance data, which is motivated by the great success of the TIWD model, shares all its positive properties, and additionally reduces the computational workload from $O(n^3)$ to $O(n^2)$ per sweep of the Gibbs sampler. Compared to vectorial representations, pairwise distances do not convey information about translations and rotations of the underlying coordinate system. While in the TIWD model this lack of information is handled by making the likelihood invariant against such geometric transformations, here we break this symmetry by choosing one particular inner-product representation $S_*$ and thus, one particular coordinate system. The advantage is being able to use a standard (i.e. central) Wishart distribution for which we present an efficient Gibbs sampling algorithm.

We show that our construction principle for selecting $S_*$ among all inner product matrices corresponding to an observed distance matrix $D$ and finds an optimal candidate if the true covariance was known. Although it is a pure theoretical guarantee, it is successfully exploited by a simple hierarchical cluster method to produce an initial covariance estimate—all without specifying the number of clusters, which is one of the model's key properties. On the algorithmic side, we prove that $S_*$ can be computed in $O(n^2)$ time using tree traversals. Assuming the number of Gibbs sweeps necessary is independent of $n$ (which, of course, depends on the problem), we now have a probabilistic algorithm for partitioning distance matrices running in $O(n^2)$ time. Experiments on simulated data show that the quality of partitions found is at least comparable to that of the original TIWD. It is now possible for the first time to use the Wishart-Dirichlet process model for large matrices. Our experiment containing $\approx 12,000$ proteins shows that fastTIWD can be successfully used to mine large relational datasets and leads to automatic identification of protein clusters sharing rare structural properties. Assuming that in most clustering problems it is acceptable to obtain a solution within some hours, any further size increase of the input matrix will become more and more a problem of memory capacity rather than computation time.

**Acknowledgments**

This work has been partially supported by the FP7 EU project SIMBAD.

**References**

[1] J. Vogt, S. Prabhakaran, T. Fuchs, and V. Roth. The Translation-invariant Wishart-Dirichlet Process for Clustering Distance Data. In *Proceedings of the 27th International Conference on Machine Learning*, 2010.

[2] P. McCullagh and J. Yang. How Many Clusters? *Bayesian Analysis*, 3:101–120, 2008.

[3] Y. W. Teh. Dirichlet Processes. In *Encyclopedia of Machine Learning*. Springer, 2010.

[4] J. Sethuraman. A Constructive Definition of Dirichlet Priors. *Statistica Sinica*, 4:639–650, 1994.

[5] B. A. Frigyik, A. Kapila, and M. R. Gupta. Introduction to the Dirichlet Distribution and Related Processes. Technical report, Departement of Electrical Engineering, University of Washington, 2010.

[6] W. Ewens. The Sampling Theory of Selectively Neutral Alleles. *Theoretical Population Biology*, 3:87–112, 1972.

[7] D. Blei and M. Jordan. Variational Inference for Dirichlet Process Mixtures. *Bayesian Analysis*, 1:121–144, 2005.

[8] R. Neal. Markov Chain Sampling Methods for Dirichlet Process Mixture Models. *Journal of Computational and Graphical Statistics*, 9(2):249–265, 2000.

[9] P. McCullagh. Marginal Likelihood for Distance Matrices. *Statistica Sinica*, 19:631–649, 2009.

[10] B. Schölkopf, A. Smola, and K.-R. Müller. Nonlinear Component Analysis as a Kernel Eigenvalue Problem. *Neural Computation*, 10(5):1299–1319, July 1998.

[11] J.A. Diaz-Garcia, J.R. Gutierrez, and K.V. Mardia. Wishart and Pseudo-Wishart Distributions and Some Applications to Shape Theory. *Journal of Multivariate Analysis*, 63:73–87, 1997.

[12] H. Uhlig. On Singular Wishart and Singular Multivariate Beta Distributions. *Annals of Statistics*, 22:395–405, 1994.

[13] M. Srivastava. Singular Wishart and Multivariate Beta Distributions. *Annals of Statistics*, 31(2):1537–1560, 2003.

[14] M. Farach, S. Kannan, and T. Warnow. A Robust Model for Finding Optimal Evolutionary Trees. In *Proceedings of the 25th Annual ACM Symposium on Theory of Computing*, pages 137–145, 1993.

